# Divisive Normalization: Justification and Effectiveness as Efficient Coding Transform

**Siwei Lyu** *
Computer Science Department
University at Albany, State University of New York
Albany, NY 12222, USA

## Abstract

*Divisive normalization (DN) has been advocated as an effective nonlinear* efficient coding *transform for natural sensory signals with applications in biology and engineering. In this work, we aim to establish a connection between the DN transform and the statistical properties of natural sensory signals. Our analysis is based on the use of multivariate* t *model to capture some important statistical properties of natural sensory signals. The multivariate* t *model justifies DN as an approximation to the transform that completely eliminates its statistical dependency. Furthermore, using the multivariate* t *model and measuring statistical dependency with multi-information, we can precisely quantify the statistical dependency that is reduced by the DN transform. We compare this with the actual performance of the DN transform in reducing statistical dependencies of natural sensory signals. Our theoretical analysis and quantitative evaluations confirm DN as an effective efficient coding transform for natural sensory signals. On the other hand, we also observe a previously unreported phenomenon that DN may increase statistical dependencies when the size of pooling is small.*

## 1 Introduction

It has been widely accepted that biological sensory systems are adapted to match the statistical properties of the signals in the natural environments. Among different ways such may be achieved, the *efficient coding hypothesis* [2, 3] asserts that a sensory system might be understood as a transform that reduces redundancies in its responses to the input sensory stimuli (e.g., odor, sounds, and time varying images). Such signal transforms, termed as *efficient coding transforms*, are also important to applications in engineering – with the reduced statistical dependencies, sensory signals can be more efficiently stored, transmitted and processed. Over the years, many works, most notably the ICA methodology, have aimed to find *linear* efficient coding transforms for natural sensory signals [20, 4, 15]. These efforts were widely regarded as a confirmation of the efficient coding hypothesis, as they lead to localized linear basis that are similar to receptive fields found physiologically in the cortex. Nonetheless, it has also been noted that there are statistical dependencies in natural images or sounds, to which linear transforms are not effective to reduce or eliminate [5, 17]. This motivates the study of *nonlinear* efficient coding transforms.

Divisive normalization (DN) is perhaps the most simple nonlinear efficient coding transform that has been extensively studied recently. The output of the DN transform is obtained from the response of a linear basis function divided by the square root of a biased and weighted sum of the squared responses of neighboring basis functions of adjacent spatial locations, orientations and scales. In biology, initial interests in DN focused on its ability to model dynamic gain control in retina [24] and the "masking" behavior in perception [11, 33], and to fit neural recordings from the mammalian

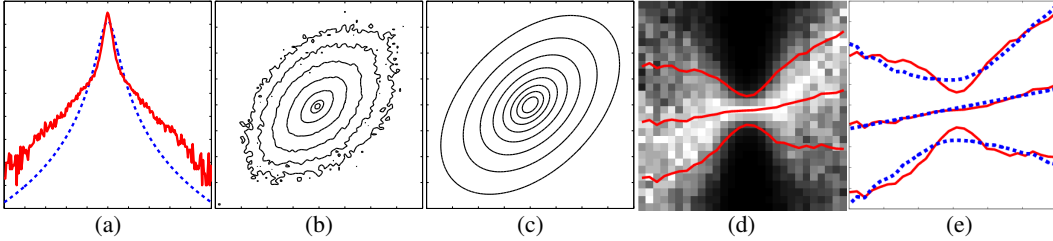

|  (a)  |  (b)  |  (c)  |  (d)  |  (e)  |

Figure 1: *Statistical properties of natural images in a band-pass domain and their representations with the multivariate* t *model.* **(a)***: Marginal densities in the log domain (images: red solid curve,* t *model: blue dashed curve).* **(b)***: Contour plot of the joint density,* $p(x_1, x_2)$*, of adjacent pairs of band-pass filter responses.* **(c)***: Contour plot of the optimally fitted multivariate* t *model of* $p(x_1, x_2)$*.* **(d)***: Each column of the image correspond to a conditional density* $p(x_1|x_2)$ *of different* $x_2$ *values.* **(e)***: The three red solid curves correspond to* $E(x_1|x_2)$ *and* $E(x_1|x_2) \pm std(x_1|x_2)$*. Blue dashed curves correspond to* $E(x_1|x_2)$ *and* $E(x_1|x_2) \pm std(x_1|x_2)$ *from the optimally fitted multivariate* t *model to* $p(x_1, x_2)$*.*

visual cortex [12, 19]. In image processing, nonlinear image representations based on DN have been applied to image compression and contrast enhancement [18, 16] showing improved performance over linear representations.

As an important nonlinear transform with such a ubiquity, it has been of great interest to find the underlying principle from which DN originates. Based on empirical observations, Schwartz and Simoncelli [23] suggested that DN can reduce statistical dependencies in natural sensory signals and is thus justified by the efficient coding hypothesis. More recent works on statistical models and efficient coding transforms of natural sensory signals (e.g., [17, 26]) have also hinted that DN may be an approximation to the optimal efficient coding transform. However, this claim needs to be rigorously validated based on statistical properties of natural sensory signals, and quantitatively evaluated with DN's performance in reducing statistical dependencies of natural sensory signals.

In this work, we aim to establish a connection between the DN transform and the statistical properties of natural sensory signals. Our analysis is based on the use of multivariate *t* model to capture some important statistical properties of natural sensory signals. The multivariate *t* model justifies DN as an approximation to the transform that completely eliminates its statistical dependency. Furthermore, using the multivariate *t* model and measuring statistical dependency with multi-information, we can precisely quantify the statistical dependency that is reduced by the DN transform. We compare this with the actual performance of the DN transform in reducing statistical dependencies of natural sensory signals. Our theoretical analysis and quantitative evaluations confirm DN as an effective efficient coding transform for natural sensory signals. On the other hand, we also observe a previously unreported phenomenon that DN may increase statistical dependencies when the size of pooling is small.

## 2 Statistical Properties of Natural Sensory Signals and Multivariate *t* Model

Sensory signals in natural environments are highly structured and non-random. Their regularities exhibit as statistical properties that distinguish them from the rest of the ensemble of all possible signals. Over the years, many distinct statistical properties of natural sensory signals have been observed. Particularly, in band-pass filtered domains where local means are removed, three statistical characteristics have been commonly observed across different signal ensembles[1]:

- symmetric and sparse non-Gaussian marginal distributions with high kurtosis [7, 10], Fig.1(a);

- joint densities of neighboring responses that have elliptically symmetric (spherically symmetric after whitening) contours of equal probability [34, 32]; Fig.1(b);

- conditional distributions of one response given neighboring responses that exhibit a "bow-tie" shape when visualized as an image [25, 6], Fig.1(d).

It has been noted that higher order statistical dependencies in the joint and conditional densities (Fig.1 (b) and (d)) cannot be effectively reduced with linear transform [17].

A compact mathematical form that can capture all three aforementioned statistical properties is the multivariate Student's $t$ model. Formally, the probability density function of a $d$ dimensional $t$ random vector $\mathbf{x}$ is defined as[2]:

$$p_t(\mathbf{x}; \alpha, \beta) = \frac{\alpha^\beta \Gamma\left(\beta + d/2\right)}{\Gamma(\beta)\sqrt{\det(\pi\Sigma)}} \left(\alpha + \mathbf{x}'\Sigma^{-1}\mathbf{x}\right)^{-\beta - d/2}, \tag{1}$$

where $\alpha > 0$ is the scale parameter and $\beta > 1$ is the shape parameter. $\Sigma$ is a symmetric and positive definite matrix, and $\Gamma(\cdot)$ is the Gamma function. From data of neighboring responses of natural sensory signals in the band-pass domain, the parameters $(\alpha, \beta)$ in the multivariate $t$ model can be obtained numerically with maximum likelihood, the details of which are given in the supplementary material.. The joint density of the fitted multivariate $t$ model has elliptically symmetric level curves of equal probability, and its marginals are 1D Student's $t$ densities that are non-Gaussian and kurtotic [14], all resembling those of the natural sensory signals, Fig.1(a) and (c). It is due to its heavy tail property that the multivariate $t$ model has been used as models of natural images [35, 22].

Furthermore, we provide another property of the multivariate $t$ model that captures the bow-tie dependency exhibited by the conditional distributions of natural sensory signals.

**Lemma 1** *Denote* $\mathbf{x}_{\backslash i}$ *as the vector formed by excluding the $i$th element from* $\mathbf{x}$. *For a $d$-dimensional isotropic* t *vector* $\mathbf{x}$ *(i.e.,* $\Sigma = I$*), we have*

$$\mathrm{E}(x_i|\mathbf{x}_{\backslash i}) = 0, \ \ and \ \ var(x_i|\mathbf{x}_{\backslash i}) = \frac{1}{2\beta + d - 3}(\alpha + \mathbf{x}'_{\backslash i}\mathbf{x}_{\backslash i}),$$

*where* $\mathrm{E}(\cdot)$ *and* $var(\cdot)$ *denote expectation and variance, respectively.*

This is proved in the supplementary material. Lemma 1 can be extended to anisotropic $t$ models by incorporating a non-diagonal $\Sigma$ using a linear "un-whitening" procedure, the result of which is demonstrated in Fig.1(e). The three red solid curves correspond to $\mathrm{E}(x_i|\mathbf{x}_{\backslash i})$ and $\mathrm{E}(x_i|\mathbf{x}_{\backslash i}) \pm \sqrt{var(x_i|\mathbf{x}_{\backslash i})}$ for pairs of adjacent band-pass filtered responses of a natural image, and the three blue dashed curves are the same quantities of the optimally fitted $t$ model. The bow-tie phenomenon comes directly from the dependencies in the conditional variances, which is precisely captured by the fitted multivariate $t$ model[3].

## 3 DN as Efficient Coding Transform for Multivariate *t* Model

Using the multivariate $t$ model as a compact representation of statistical properties of natural sensory signals in linear band-pass domains, our aim is to find an efficient coding transform that can effectively reduce its statistical dependencies. This is based on an important property of the multivariate $t$ model – it is a special case of the *Gaussian scale mixture* (GSM) [1]. More specifically, the joint density $p_t(\mathbf{x}; \alpha, \beta)$ can be written as an infinite mixture of Gaussians with zero mean and covariance matrix $\Sigma$, as

$$p_t(\mathbf{x}; \alpha, \beta) = \int_0^\infty \frac{1}{\sqrt{\det(2\pi z\Sigma)}} \exp\left(-\frac{1}{2z}\mathbf{x}'\Sigma^{-1}\mathbf{x}\right) p_{\gamma^{-1}}(z; \alpha, \beta)dz,$$

where $p_{\gamma^{-1}}(z) = \frac{\alpha^\beta}{2^\beta \Gamma(\beta)} z^{-\beta - 1} \exp\left(-\frac{\alpha}{2z}\right)$ is the *inverse Gamma distribution*. Equivalently, for a $d$ dimensional $t$ vector $\mathbf{x}$, we can decompose it into the product of two independent variables $\mathbf{u}$ and $\sqrt{z}$, as $\mathbf{x} = \mathbf{u} \cdot \sqrt{z}$, where $\mathbf{u}$ is a $d$-dim Gaussian vector with zero mean and covariance matrix $\Sigma$, and $z > 0$ is a scalar variable of an inverse Gamma law with parameter $(\alpha, \beta)$. To simplify the discussion, hereafter we will assume that the signals have been whitened so that there is no second-order dependencies in $\mathbf{x}$. Correspondingly, the Gaussian vector $\mathbf{u}$ has a covariance $\Sigma = I$.

According to the GSM equivalence of the multivariate $t$ model, we have $\mathbf{u} = \mathbf{x}/\sqrt{z}$. As an isotropic Gaussian vector has mutually independent components, there is no statistical dependency among elements of $\mathbf{u}$. In other words, $\mathbf{x}/\sqrt{z}$ equals to a transform that completely eliminates all statistical dependencies in $\mathbf{x}$. Unfortunately, this optimal efficient coding transform is not realizable, because $z$ is a latent variable that we do not have direct access to.

To overcome this difficulty, we can use an estimator of $z$ based on the visible data vector $\mathbf{x}$, $\hat{z}$, to approximate the true value of $z$, and obtain an approximation to the optimal efficient coding

transform as $\mathbf{x}/\sqrt{\hat{z}}$. For the multivariate $t$ model, it turns out that two most common choices for the estimators $z$, namely, the *maximum a posterior* (MAP) and the *Bayesian least square* (BLS) estimators, and a third estimator all have similar forms, a result formally stated in the following lemma (a proof is given in the supplementary material).

**Lemma 2** *For the $d$-dimensional isotropic* t *vector* $\mathbf{x}$ *with parameters* $(\alpha, \beta)$, *we consider three estimators of $z$ as: (i) the MAP estimator,* $\hat{z}_1 = \arg\max_z p(z|\mathbf{x})$, *which is the mode of the posterior density, (ii) the BLS estimator, which is the mean of the posterior density* $\hat{z}_2 = \mathrm{E}_{z|\mathbf{x}}(z|\mathbf{x})$, *and (iii) the inverse of the conditional mean of $1/z$, as* $\hat{z}_3 = \left(\mathrm{E}_{z|\mathbf{x}}(1/z|\mathbf{x})\right)^{-1}$, *which are:*

$$\hat{z}_1 = \frac{\alpha + \mathbf{x}'\mathbf{x}}{2\beta + d + 2}, \quad \hat{z}_2 = \frac{\alpha + \mathbf{x}'\mathbf{x}}{2\beta + d - 2}, \quad and \quad \hat{z}_3 = \left(E_{z|\mathbf{x}}(1/z|\mathbf{x})\right)^{-1} = \frac{\alpha + \mathbf{x}'\mathbf{x}}{2\beta + d}.$$

If we drop the irrelevant scaling factors from each of these estimators and plug them in $\mathbf{x}/\sqrt{\hat{z}}$, we obtain a nonlinear transform of $\mathbf{x}$ as,

$$\mathbf{y} = \phi(\mathbf{x}), \text{ where } \phi(\mathbf{x}) \equiv \frac{\mathbf{x}}{\sqrt{\alpha + \mathbf{x}'\mathbf{x}}} = \frac{\|\mathbf{x}\|}{\sqrt{\alpha + \|\mathbf{x}\|^2}} \frac{\mathbf{x}}{\|\mathbf{x}\|}. \tag{2}$$

This is the standard form of *divisive normalization* that will be used throughout this paper. Lemma 2 shows that the DN transform is justified as an approximate to the optimal efficient coding transform given a multivariate $t$ model of natural sensory signals. Our result also shows that the DN transform approximately "gaussianizes" the input data, a phenomenon that has been empirically observed by several authors (e.g., [6, 23]).

## 3.1 Properties of DN Transform

The standard DN transform given by Eq.(2) has some nice and important properties. Particularly, the following Lemma shows that it is invertible and its Jacobian determinant has closed form.

**Lemma 3** *For the standard DN transform given in Eq. (2), its inversion for* $\mathbf{y} \in R^d$ *with* $\|\mathbf{y}\| < 1$ *is* $\phi^{-1}(\mathbf{y}) = \frac{\sqrt{\alpha}\mathbf{y}}{\sqrt{1-\|\mathbf{y}\|^2}} = \frac{\sqrt{\alpha}\|\mathbf{y}\|}{\sqrt{1-\|\mathbf{y}\|^2}} \frac{\mathbf{y}}{\|\mathbf{y}\|}$. *The determinant of its Jacobian matrix is also in closed form, which is given by* $\det\left(\frac{\partial\phi(\mathbf{x})}{\partial\mathbf{x}}\right) = \alpha(\alpha + \mathbf{x}'\mathbf{x})^{-(d/2+1)}$.

Further, the DN transform of a multivariate $t$ vector also has a closed form density function.

**Lemma 4** *If* $\mathbf{x} \in R^d$ *has an isotropic* t *density with parameter* $(\alpha, \beta)$, *then its DN transform,* $\mathbf{y} = \phi(\mathbf{x})$, *follows an isotropic r model, whose probability density function is*

$$p_\tau(\mathbf{y}) = \begin{cases} \frac{\Gamma(\beta+d/2)}{\pi^{d/2}\Gamma(\beta)}\left(1 - \mathbf{y}'\mathbf{y}\right)^{\beta-1} & \|\mathbf{y}\| < 1 \\ 0 & \|y\| \geq 1 \end{cases} \tag{3}$$

Lemma 4 suggests a duality between $t$ and r models with regards to the DN transform. Proofs of Lemma 3 and Lemma 4 can be found in [8]. For completeness, we also provide our proofs in the supplementary material.

## 3.2 Equivalent Forms of DN Transform

In the current literature, the DN transform has been defined in many different forms other than Eq.(2). However, if we are merely interested in their ability to reduce statistical dependencies, many of the different forms of DN transform based on $l_2$ norm of the input vector $\mathbf{x}$ become equivalent. To be more specific, we quantify statistical statistical dependency of a random vector $\mathbf{x}$ using the multi-information (MI) [27], defined as

$$I(\mathbf{x}) = \int_{\mathbf{x}} p(\mathbf{x}) \log\left(p(\mathbf{x})/\prod_{k=1}^{d} p(x_k)\right) d\mathbf{x} = \sum_{k=1}^{d} H(x_k) - H(\mathbf{x}), \tag{4}$$

where $H(\cdot)$ denotes the Shannon differential entropy. MI is non-negative, and is zero if and only if the components of $\mathbf{x}$ are mutually independent. MI is a generalization of mutual information, and the two become identical when measures dependency for two dimensional $\mathbf{x}$. Furthermore, MI is invariant to any operation that operates on individual components of $\mathbf{x}$ (e.g., element-wise rescaling) since such operations produce an equal effect on the two terms $\sum_{k=1}^{d} H(x_k)$ and $H(\mathbf{x})$ (see [27]).

Now consider four different definitions of the DN transform expressed in terms of the individual element of the output vector as

$$y_i = \frac{x_i}{\sqrt{\alpha + \mathbf{x}'\mathbf{x}}}, \quad s_i = \frac{x_i^2}{\alpha + \mathbf{x}'\mathbf{x}}, \quad v_i = \frac{x_i}{\sqrt{\alpha + \mathbf{x}'_{\backslash i}\mathbf{x}_{\backslash i}}}, \quad t_i = \frac{x_i^2}{\alpha + \mathbf{x}'_{\backslash i}\mathbf{x}_{\backslash i}}.$$

Here $\mathbf{x}_{\backslash i}$ denotes the vector formed from $\mathbf{x}$ without its $i$th component. Specifically, $y_i$ is the output of Eq.(2). $s_i$ is the output of the original DN transform used by Heeger [12]. $v_i$ corresponds to the DN transform used by Schwartz and Simoncelli [23]. The main difference with Eq.(2) is that the denominator is formed without element $x_i$. Last, $t_i$ is the output of the DN transform used in [31]. These forms of DN[4] related with each other by element-wise operations, as we have

$$s_i = y_i^2, \quad v_i = \frac{x_i}{\sqrt{\alpha + \mathbf{x}'_{\backslash i}\mathbf{x}_{\backslash i}}} = \frac{x_i}{\sqrt{\alpha + \mathbf{x}'\mathbf{x} - x_i^2}} = \frac{y_i}{\sqrt{1 - y_i^2}}, \quad \text{and} \quad t_i = s_i^2 = \frac{y_i^2}{1 - y_i^2}.$$

As element-wise operations do not affect MI, in terms of dependency reduction, all three transforms are equivalent to the standard form in terms of reducing statistical dependencies. Therefore, the subsequent analysis applies to all these equivalent forms of the DN transform.

## 4 Quantifying DN Transform as Efficient Coding Transform

We have set up a relation between the DN transform with statistical properties of natural sensory signals through the multivariate $t$ model. However, its effectiveness as an efficient coding transform for natural sensory signals needs yet to be quantified for two reasons. First, DN is only an approximation to the optimal transform that eliminates statistical dependencies in a multivariate $t$ model. Further, the multivariate $t$ model itself is a surrogate of the true statistical model of natural sensory signals. It is our goal in this section to quantify the effectiveness of the DN transform in reducing statistical dependencies. We start with a study of applying DN to the multivariate $t$ model, the closed form density of which permits us a theoretical analysis of DN's performance in dependency reduction. We then appy DN to real natural sensory signal data, and compare its effectiveness as an efficient coding transform with the theoretical prediction obtained with the multivariate $t$ model.

### 4.1 Results with Multivariate $t$ Model

For simplicity, we consider isotropic models whose second order dependencies are removed with whitening. The density functions of multivariate $t$ and r models lead to closed form solutions for MI, as formally stated in the following lemma (proved in the supplementary material).

**Lemma 5** *The MI of a $d$-dimensional isotropic* t *vector* $\mathbf{x}$ *is*

$$
\begin{aligned}
I(\mathbf{x}) &= (d-1)\log\Gamma(\beta) - d\log\Gamma(\beta + 1/2) + \log\Gamma(\beta + d/2) - (d-1)\beta\Psi(\beta) \\
&\quad + d(\beta + 1/2)\Psi(\beta + 1/2) - (\beta + d/2)\Psi(\beta + d/2).
\end{aligned}
$$

*Similarly, the MI of a $d$-dimensional* r *vector* $\mathbf{y} = \phi(\mathbf{x})$, *which is the DN transform of* $\mathbf{x}$, *is*

$$
\begin{aligned}
I(\mathbf{y}) &= d\log\Gamma(\beta + (d-1)/2) - \log\Gamma(\beta) - (d-1)\log\Gamma(\beta + d/2) + (\beta - 1)\Psi(\beta) \\
&\quad + (d-1)(\beta + d/2 - 1)\Psi(\beta + d/2) - d(\beta + (d-3)/2)\Psi(\beta + (d-1)/2).
\end{aligned}
$$

*In both cases,* $\Psi(\beta)$ *denotes the Digamma function which is defined as* $\Psi(\beta) = \frac{d}{d\beta}\log\Gamma(\beta)$.

Note that $\alpha$ does not appear in these formulas, as it can be removed by re-scaling data and has no effect on MI. Using Lemma 5, for a $d$-dimensional $t$ vector, if we have $I(\mathbf{x}) > I(\mathbf{y})$, the DN transform reduces its statistical dependency, conversely, if $I(\mathbf{x}) < I(\mathbf{y})$, it increases dependency. As both Gamma function and Digamma function can be computed to high numerical precision, we can evaluate $\Delta I = I(\mathbf{x}) - I(\mathbf{y})$ corresponding to different shape parameter $\beta$ and data dimensionality $d$. The left panel of Fig.2 illustrates the surface of $\Delta I/I(\mathbf{x})$, which measures the relative change in MI between an isotropic $t$ vector and its DN transform. The right panel of Fig.2 shows one dimensional curves of $\Delta I/I(\mathbf{x})$ corresponding to different $d$ values with varying $\beta$.

These plots illustrate several interesting aspects of the DN transform as an approximate efficient coding transform of the multivariate $t$ models. First, with data dimensionality $d > 4$, using DN

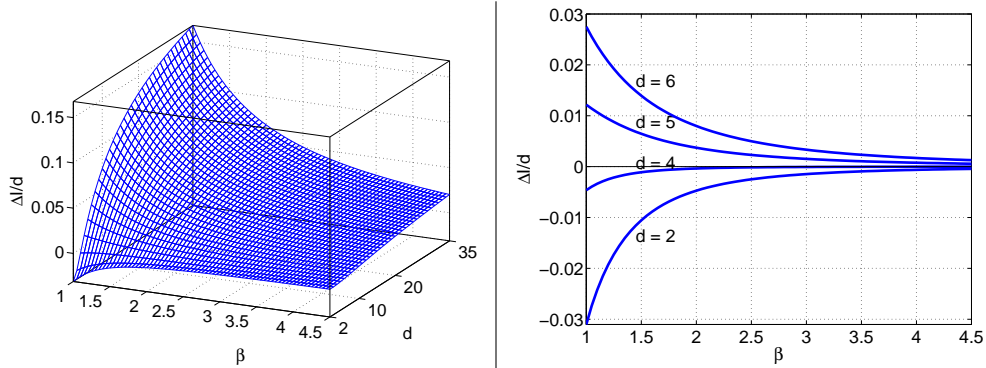

Figure 2: **left**: *Surface plot of $[I(\mathbf{x}) - I(\phi(\mathbf{x}))]/I(\mathbf{x})$, measuring MI changes after applying the DN transform $\phi(\cdot)$ to an isotropic* t *vector* $\mathbf{x}$. $I(\mathbf{x})$ *and* $I(\phi(\mathbf{x}))$ *computed numerically using Lemma 5. The two coordinates correspond with data dimensionality (d) and shape parameters of the multivariate* t *model ($\beta$).* **right**: *one dimensional curves of $\Delta I/I(\mathbf{x})$ corresponding to different d values with varying $\beta$.*

leads to significant reduction of statistical dependency, but such reductions become weaker as $\beta$ increases. On the other hand, our experiment also showed an unexpected behavior that has not been reported before, for $d \leq 4$, the change of MI caused by the use of DN is negative, i.e., DN *increases* statistical dependency for such cases. Therefore, though effective for high dimensional models, DN is *not* an efficient coding transform for low dimensional multivariate *t* models.

## 4.2 Results with Natural Sensory Signals

As mentioned previously, the multivariate *t* model is an approximation to the source model of natural sensory signals. Therefore, we would like to compare our analysis in the previous section with the actual dependency reduction performance of the DN transform on real natural sensory signal data.

### 4.2.1 Non-parametric Estimating MI Changes

To this end, we need to evaluate MI changes after applying DN without relying on any specific parametric density model. This has been achieved previously for two dimensional data using straightforward nonparametric estimation of MI based on histograms [28]. However, the estimations obtained this way are prone to strong bias due to the binning scheme in generating the histograms [21], and cannot be generalized to higher data dimensions due to the "curse of dimensionality", as the number of bins increases exponentially with regards to the data dimension.

Instead, in this work, we directly compute the *difference* of MI after DN is applied without explicitly binning data. To see how this is possible, we first express the computation of the MI change as

$$I(\mathbf{x}) - I(\mathbf{y}) = \sum_{k=1}^{d} H(x_k) - \sum_{k=1}^{d} H(y_k) - H(\mathbf{x}) + H(\mathbf{y}). \tag{5}$$

Next, the entropy of $\mathbf{y} = \phi(\mathbf{x})$ is related to the entropy of $\mathbf{x}$, as $H(\mathbf{y}) = H(\mathbf{x}) - \int_{\mathbf{x}} p(\mathbf{x}) \log \left| \det \left( \frac{\partial \phi(\mathbf{x})}{\partial \mathbf{x}} \right) \right| d\mathbf{x}$, where $\det \left( \frac{\partial \phi(\mathbf{x})}{\partial \mathbf{x}} \right)$ is the Jacobian determinant of $\phi(\mathbf{x})$ [9]. For DN, $\det \left( \frac{\partial \phi(\mathbf{x})}{\partial \mathbf{x}} \right)$ has closed form (Lemma 3), and replacing it in Eq.(5) yields

$$I(\mathbf{y}) - I(\mathbf{x}) = \sum_{k=1}^{d} H(y_k) - \sum_{k=1}^{d} H(x_k) + \log \alpha - \left( \frac{d}{2} + 1 \right) \int_{\mathbf{x}} p(\mathbf{x}) \log (\alpha + \mathbf{x}'\mathbf{x}) d\mathbf{x}. \tag{6}$$

Once we determine $\alpha$, the last term in Eq.(6) can be approximated with the average of function $\log (\alpha + \mathbf{x}'\mathbf{x})$ over input data. The first two terms requires direct estimation of differential entropies of scalar random variables, $H(y_k)$ and $H(x_k)$. For a more reliable estimation, we use the nonparametric "bin-less" *m-spacing estimator* [30]. As a simple sanity check, Fig.3(a) shows the theoretical evaluation of $(I(\mathbf{y}) - I(\mathbf{x}))/d$ obtained with Lemma 5 for isotropic *t* models with $\beta = 1.10$ and varying $d$ (blue solid curve). The red dashed curve shows the same quantity computed using Eq.(6) with $10,000$ random samples drawn from the same multivariate *t* models. The small difference between the two curves in this plot confirms the quality of the non-parametric estimation.

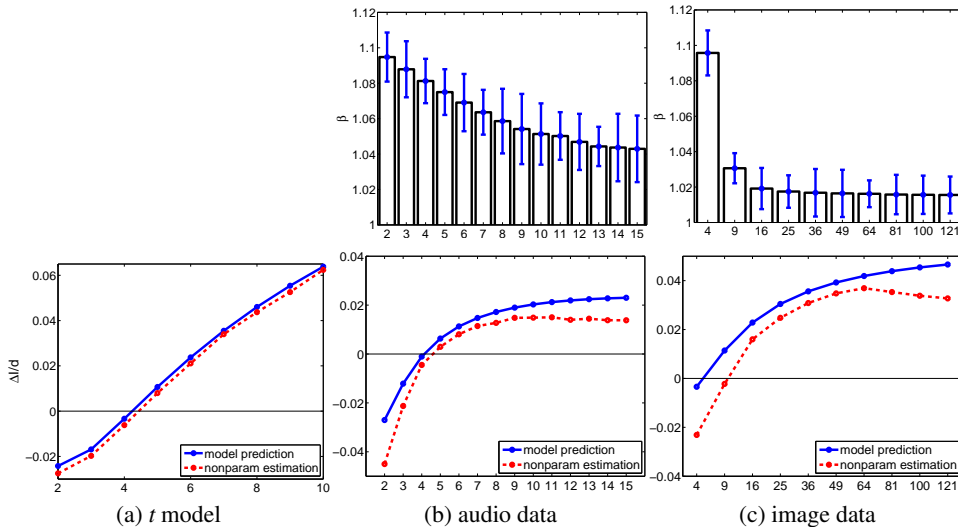

(a) t model          (b) audio data          (c) image data

Figure 3: **(a)** *Comparison of theoretical prediction of MI reduction for isotropic* t *model with $\beta = 1.1$ and different dimensions (blue solid curve) with the non-parametric estimation using Eq.(6) and $m$-spacing estimator [30] on $10,000$ random samples drawn from the corresponding multivariate* t *models (red dashed curve).* **(b)** *Top row is the mean and standard deviation of the estimated shape parameter $\beta$ for natural audio data of different local window sizes. Bottom row is the comparison of MI changes ($\Delta I/d$). Blue solid curve corresponds to the prediction with Lemma 5, red dashed curve is the non-parametric estimation of Eq.(6).* **(c)** *Same results as (b) for natural image data with different local block sizes.*

### 4.2.2 Experimental Evaluation and Comparison

We next experiment with natural audio and image data. For audio, we used 20 sound clips of animal vocalization and recordings in natural environments, which have a sampling frequency of 44.1 kHz and typical length of $15 - 20$ seconds. These sound clips were filtered with a bandpass gamma-tone filter of 3 kHz center frequency [13]. For image data, we used eight images in the van Hateren database [29]. These images have contents of natural scenes such as woods and greens with linearized intensity values. Each image was first cropped to the central $1024 \times 1024$ region and then subject to a log transform. The log pixel intensities are further adjusted to have a zero mean. We further processed the log transformed pixel intensities by convolving with an isotropic bandpass filter that captures an annulus of frequencies in the Fourier domain ranging from $\pi/4$ to $\pi$ radians/pixel. Finally, data used in our experiments are obtained by extracting adjacent samples in localized 1D temporal (for audios) or 2D spatial (for images) windows of different sizes. We further whiten the data to remove second order dependencies.

With these data, we first fit multivariate *t* models using maximum likelihood (detailed procedure given in the supplementary material), from which we compute the theoretical prediction of MI difference using Lemma 5. Shown in the top row of Fig.3 (b) and (c) are the means and standard deviations of the estimated shape parameters of different sizes of local windows for audio and image data, respectively. These plots suggest two properties of the fitted multivariate *t* model. First, the estimated $\beta$ values are typically close to one due to the high kurtosis of these signal ensembles. Second, the shape parameter in general decreases as the data dimension increases.

Using the same data, we obtain the optimal DN transform by searching for optimal $\alpha$ in Eq.(2) that maximizes the change in MI given by Eq.(6). However, as entropy is estimated non-parametrically, we cannot use gradient based optimization for $\alpha$. Instead, with a range of possible $\alpha$ values, we perform a binary search, at each step of which we evaluate Eq.(6) using the current $\alpha$ and the nonparametric estimation of entropy based on the data set.

In the bottom rows of Fig.3 (b) (for audios) and (c) (for images), we show MI changes of using DN on natural sensory data that are predicted by the optimally fitted *t* model (blue solid curves) and that obtained with optimized DN parameters using nonparametric estimation of Eq.(6) (red dashed curve). For robustness, these results are the averages over data sets from the 20 audio signals and 8 images, respectively. In general, changes in statistical dependencies obtained with the optimal DN transforms are in accordance with those predicted by the multivariate *t* model. The model-

based predictions also tend to be upper-bounds of the actual DN performance. Some discrepancies between the two start to show as dimensionality increases, as the dependency reductions achieved with DN become smaller even though the model-based predictions tend to keep increasing. This may be caused by the approximation nature of the multivariate $t$ model to natural sensory data. As such, more complex structures in the natural sensory signals, especially with larger local windows, cannot be effectively captured by the multivariate $t$ models, which renders DN less effective.

On the other hand, our observation based on the multivariate $t$ model that the DN transform tends to increase statistical dependency for small pooling size also holds to real data. Indeed, the increment of MI becomes more severe for $d \leq 4$. On the surface, our finding seems to be in contradiction with [23], where it was empirically shown that applying an equivalent form of the DN transform as Eq.(2) (see Section 3.2) over a pair of input neurons can reduce statistical dependencies. However, one key yet subtle difference is that statistical dependency is defined as the correlations in the conditional variances in [23], i.e., the bow-tie behavior as in Fig.1(d). The observation made in [23] is then based on the empirical observations that after applying DN transform, such dependencies in the transformed variables become weaker, while our results show that the statistical dependency measured by MI in that case actually *increases*.

## 5   Conclusion

In this work, based on the use of the multivariate $t$ model of natural sensory signals, we have presented a theoretical analysis showing that DN emerges as an approximate efficient coding transform. Furthermore, we provide a quantitative analysis of the effectiveness of DN as an efficient coding transform for the multivariate $t$ model and natural sensory signal data. These analyses confirm the ability of DN in reducing statistical dependency of natural sensory signals. More interestingly, we observe a previously unreported result that DN can actually increase statistical dependency when the size of pooling is small. As a future direction, we would like to extend this study to a generalized DN transform where the denominator and numerator can have different degrees.

**Acknowledgement** The author would like to thank Eero Simoncelli for helpful discussions, and the three anonymous reviewers for their constructive comments.

## Footnotes

*This work is supported by an NSF CAREER Award (IIS-0953373).

[1]The results in Fig.1 are obtained with spatial neighbors in images. Similar behaviors have also been observed for orientation and scale neighbors [6], as well as other type of sensory signals such as audios [23, 17].

[2]Eq.(1) can be shown to be equivalent to the standard definition of multivariate $t$ density in [14].

[3]The dependencies illustrated are nonlinear because we use conditional standard deviations.

[4]There are usually weights to each $x_i^2$ in the denominator, but re-scaling data can remove the different weights and leads to no change in terms of MI.

## References

[1] D. F. Andrews and C. L. Mallows. Scale mixtures of normal distributions. *Journal of the Royal Statistical Society. Series B (Methodological)*, 36(1):99–102, 1974.

[2] F Attneave. Some informational aspects of visual perception. *Psych. Rev.*, 61:183–193, 1954.

[3] H B Barlow. Possible principles underlying the transformation of sensory messages. In W A Rosenblith, editor, *Sensory Communication*, pages 217–234. MIT Press, Cambridge, MA, 1961.

[4] A J Bell and T J Sejnowski. The 'independent components' of natural scenes are edge filters. 37(23):3327–3338, 1997.

[5] Matthias Bethge. Factorial coding of natural images: how effective are linear models in removing higher-order dependencies? *J. Opt. Soc. Am. A*, 23(6):1253–1268, 2006.

[6] R. W. Buccigrossi and E. P. Simoncelli. Image compression via joint statistical characterization in the wavelet domain. 8(12):1688–1701, 1999.

[7] P.J. Burt and E.H. Adelson. The Laplacian pyramid as a compact image code. *IEEE Transactions on Communication*, 31(4):532–540, 1981.

[8] J. Costa, A. Hero, and C. Vignat. On solutions to multivariate maximum $\alpha$-entropy problems. In *EMM-CVPR*, 2003.

[9] T. Cover and J. Thomas. *Elements of Information Theory*. Wiley-Interscience, 2nd edition, 2006.

[10] D J Field. Relations between the statistics of natural images and the response properties of cortical cells. 4(12):2379–2394, 1987.

[11] J. Foley. Human luminence pattern mechanisims: Masking experimants require a new model. *J. of Opt. Soc. of Amer. A*, 11(6):1710–1719, 1994.

[12] D. J. Heeger. Normalization of cell responses in cat striate cortex. *Visual neural science*, 9:181–198, 1992.

[13] P. Johannesma. The pre-response stimulus ensemble of neurons in the cochlear nucleus. In *Symposium on Hearing Theory*, pages 58–69, Eindhoven, Holland, 1972.

[14] Samuel Kotz and Saralees Nadarajah. *Multivariate t Distributions and Their Applications*. Cambridge University Press, 2004.

[15] M S Lewicki. Efficient coding of natural sounds. *Nature Neuroscience*, 5(4):356–363, 2002.

[16] S. Lyu and E. P. Simoncelli. Nonlinear image representation using divisive normalization. In *IEEE Conference on Computer Vision and Patten Recognition (CVPR)*, Anchorage, AK, June 2008.

[17] S Lyu and E P Simoncelli. Nonlinear extraction of 'independent components' of natural images using radial Gaussianization. *Neural Computation*, 18(6):1–35, 2009.

[18] J. Malo, I. Epifanio, R. Navarro, and E. P. Simoncelli. Non-linear image representation for efficient perceptual coding. 15(1):68–80, January 2006.

[19] V. Mante, V. Bonin, and M. Carandini. Functional mechanisms shaping lateral geniculate responses to artificial and natural stimuli. *Neuron*, 58:625–638, May 2008.

[20] B A Olshausen and D J Field. Emergence of simple-cell receptive field properties by learning a sparse code for natural images. *Nature*, 381:607–609, 1996.

[21] Liam Paninski. Estimation of entropy and mutual information. *Neural Comput.*, 15(6):1191–1253, 2003.

[22] S. Roth and M. Black. Fields of experts: A framework for learning image priors. volume 2, pages 860–867, 2005.

[23] O. Schwartz and E. P. Simoncelli. Natural signal statistics and sensory gain control. *Nature Neuroscience*, 4(8):819–825, August 2001.

[24] R Shapley and C Enroth-Cugell. Visual adaptation and retinal gain control. *Progress in Retinal Research*, 3:263–346, 1984.

[25] E P Simoncelli and R W Buccigrossi. Embedded wavelet image compression based on a joint probability model. In *Proc 4th IEEE Int'l Conf on Image Proc*, volume I, pages 640–643, Santa Barbara, October 26-29 1997. IEEE Sig Proc Society.

[26] Fabian H. Sinz and Matthias Bethge. The conjoint effect of divisive normalization and orientation selectivity on redundancy reduction. In *NIPS*. 2009.

[27] M. Studeny and J. Vejnarova. The multiinformation function as a tool for measuring stochastic dependence. In M. I. Jordan, editor, *Learning in Graphical Models*, pages 261–297. Dordrecht: Kluwer., 1998.

[28] Roberto Valerio and Rafael Navarro. Input–output statistical independence in divisive normalization models of v1 neurons. *Network: Computation in Neural Systems*, 14(4):733–745, 2003.

[29] A van der Schaaf and J H van Hateren. Modelling the power spectra of natural images: Statistics and information. *Vision Research*, 28(17):2759–2770, 1996.

[30] Oldrich Vasicek. A test for normality based on sample entropy. *Journal of the Royal Statistical Society, Series B*, 38(1):54–59, 1976.

[31] M. J. Wainwright, O. Schwartz, and E. P. Simoncelli. Natural image statistics and divisive normalization: Modeling nonlinearity and adaptation in cortical neurons. In *Probabilistic Models of the Brain: Perception and Neural Function*, pages 203–222. MIT Press, 2002.

[32] M J Wainwright and E P Simoncelli. Scale mixtures of Gaussians and the statistics of natural images. In S. A. Solla, T. K. Leen, and K.-R. Müller, editors, *Adv. Neural Information Processing Systems (NIPS*99)*, volume 12, pages 855–861, Cambridge, MA, May 2000. MIT Press.

[33] A. Watson and J. Solomon. A model of visual contrast gain control and pattern masking. *J. Opt. Soc. Amer. A*, pages 2379–2391, 1997.

[34] B Wegmann and C Zetzsche. Statistical dependence between orientation filter outputs used in an human vision based image code. In *Proc Visual Comm. and Image Processing*, volume 1360, pages 909–922, Lausanne, Switzerland, 1990.

[35] M. Welling, G. E. Hinton, and S. Osindero. Learning sparse topographic representations with products of Student-t distributions. pages 1359–1366, 2002.

